# Breaking Boundaries: Active Information Acquisition Across Learning and Diagnosis

**Ashish Kapoor and Eric Horvitz**
Microsoft Research
1 Microsoft Way
Redmond, WA 98052

## Abstract

To date, the processes employed for active information acquisition during periods of learning and diagnosis have been considered as separate and have been applied in distinct phases of analysis. While active learning centers on the collection of information about training cases in order to build better predictive models, diagnosis uses fixed predictive models for guiding the collection of observations about a specific test case at hand. We introduce a model and inferential methods that bridge these phases of analysis into a holistic approach to information acquisition that considers simultaneously the extension of the predictive model and the probing of a case at hand. The bridging of active learning and real-time diagnostic feature acquisition leads to a new class of policies for learning and diagnosis.

## 1 Introduction

Consider a real-world problem scenario where the challenge is to diagnose a patient who presents with several salient symptoms by performing inference with a probabilistic diagnostic model. The diagnostic model is trained from a database of patients, where training cases may have missing features. Assume we have at our discretion an evidential budget that enables us to acquire additional information so as to make a good diagnosis. Traditionally, such a budget has been spent solely on performing real-time observations about the case at hand, for example, by carrying out additional tests on a patient presenting to a physician with some previously identified complaints, signs, and symptoms. However, there lies another opportunity to improving diagnostic models—that of allocating some or all of the evidential budget to extending some portion of the training database, and then learning an updated diagnostic model for use in inference about the case at hand. This broader perspective on diagnostic reasoning has real-world implications. For instance, investing efforts to observe features that are currently missing in training cases, such as missing details on presenting symptoms or on outcomes of prior patient cases, might preempt the need for carrying out a painful or risky medical test on the patient at hand. We focus on the promise of developing methods that jointly consider informational value and costs of acquiring information about both the case at hand and about cases in the training library, and weighing the potential contributions of each of these potential sources of information during diagnosis.

To date, the process of diagnosis has focused on the use of a *fixed* predictive model, which in turn is used to generate recommendations for the observations to gather. Similarly, efforts in active learning have focused on gathering information about the training cases in order to build better predictive models. The active collection of the different types of missing information under a budget, spanning methods that have been referred to separately as *learning* and *diagnosis*, is graphically depicted in Figure 1. While *diagnosis-time* information acquisition methods focus on acquiring information about the test case at hand, *induction-time* methods focus on collecting information about training cases for learning a good predictive model. We shall describe methods that weave together these two perspectives on information acquisition that have been handled separately to date, yielding a holistic approach to evidence collection in the context of the larger learning and prediction system. The

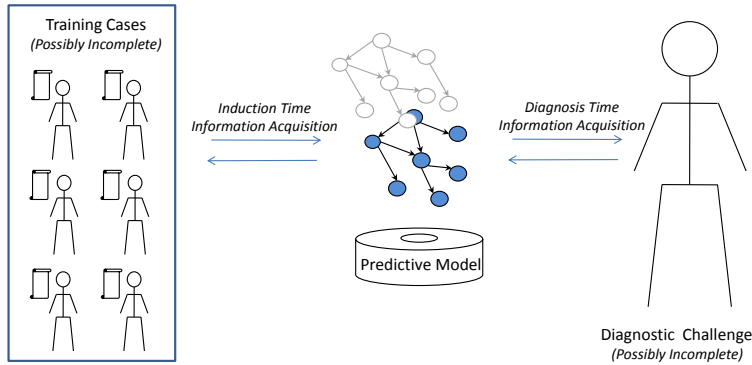

Figure 1: Illustration of induction-time and diagnosis-time active information acquisition. Induction-time active learning focuses on acquiring information for the pool of data used to train a diagnostic model; diagnosis-time information acquisition focuses on the next best observations to acquire from the test case at hand.

methodology applies to situations where there is a single diagnostic challenge, as well as broader conceptions of diagnosis over streams of cases over time.

We take a decision-theoretic perspective on the joint consideration of observations about the case at hand and about options for extending the training set. We start by directly modeling how the training data might affect the outcome of the predictions about test cases at hand, thus, relaxing the common assumption that a predictive model is fixed during diagnosis. Real-world diagnostic applications have made this assumption to date, often employing an information-theoretic or decision theoretic-criterion, such as *value of information* (VOI), during diagnosis to collect data about the case at hand. The holistic method can guide the acquisition of data for training cases that are missing arbitrary combinations of features and labels. The methodology extends active learning beyond the situation where training is done from a case library of completely specified instances, where each case contains a complete set of observations. We shall show how the more holistic active-learning approach allows for a fine-grained triaging of information to acquire by deliberating in parallel about the value of acquiring missing information from cases either in the training or the test set.

## 2   Related Research

As we mentioned, efforts to date on the use of active learning for training classification models have largely focused on the task of acquiring labels, and assume that all of the features are observed in advance. Popular heuristics for selecting unlabeled data points include uncertainty in classification [1, 2], reduction in version space for SVMs [13], expected informativeness [9], disagreement among a committee of classifiers [3], and expected reduction in classification [10]. There has been limited work on methods for actively selecting missing features for instantiation. Lizotte et al. [8] tackle the problem of selecting features in a budgeted learning scenario. Specifically, they solve a problem that can be viewed as the inverse of traditional active learning; given class labels, they seek to determine the best features to compute for each instance such that a good predictive model can be trained under a budget. Even rarer are attempts to unify active acquisition of features with the acquisition of missing class labels. Research on this more general active learning includes work with graphical probabilistic models by Tong and Koller [14] and by Saar-Tsechansky et al. [11].

Several methods have been used for guiding data acquisition at diagnosis time. The goal is to identify the best additional observations to acquire for making inferences and for ultimately taking actions given inferences about the class of a test case at hand [4, 5, 6, 7, 12]. The best tests and observations to make are computed with methods that compute or approximate the VOI. VOI for each potential new observation is computed by considering the probability distribution over the class of the case at focus of attention of based on observations made so far, and the uncertainties expected after making each proposed observation. New evidence to collect is triaged by considering the expected utility of the best immediate actions versus the actions taken after the new observations, considering the

costs of making each proposed observation. Thus, VOI balances the informational benefits and the observational costs of the new observations under uncertainty.

# 3 Approach

We shall now describe a Bayesian model that smoothly combines induction-time and diagnosis-time information acquisition. The methods move beyond the task of parameter and structure estimation explored in the prior studies of active learning and directly model statistical relationships amongst the data points.

Assume that we are given a training corpus with $n$ independent training instances $\mathcal{D}_i = \{(\mathbf{x}_i, t_i)\}$. Here, $\mathbf{x}_i$ are the $d$ dimensional features and their labels are denoted as $t_i$. The training cases can be incomplete; not all of the labels and features in the training set $\mathcal{D}$ are observed. Hence, we represent $\mathcal{D}_i = \mathcal{D}_i^o \bigcup \mathcal{D}_i^h$, where $\mathcal{D}_i^o$ and $\mathcal{D}_i^h$ represent the mutually exclusive subsets of observed and unobserved components respectively in the $i^{th}$ data instance.

Let us consider a test data point as $\mathbf{x}_*$ where our task is to recover the label $t_*$ for the test case[1]. Similar to the training cases, we again assume that $\mathbf{x}_*$ is not fully observed and that there are unobserved features. Given a budget for acquiring information, our goal is to determine the missing components either from the training set or among the missing features in the test case so that we make the best prediction on $t_*$.

Approaches to active learning leverage the statistical relationships among sets of observations within cases with their class labels. The computation of expected value of information has been carried out with an information-theoretic method such as with procedures that seek to minimize entropy or maximize information gain. We compute such measures by directly modeling the conditional density of the test label $t^*$, given all that has been observed:

$$p(t_*|\mathbf{x}_*^o, \mathcal{D}^o) = p(t_*|\mathbf{x}_*^o, \mathcal{D}_1^o, .., \mathcal{D}_n^o) \tag{1}$$

Here, $\mathbf{x}_*^o$ represents the observed components of the test case and we define the set of all observed variables in the training corpus as $\mathcal{D}^o = \{\mathcal{D}_1^o, .., \mathcal{D}_n^o\}$ (similarly we'll use $\mathcal{D}^h = \{\mathcal{D}_1^h, .., \mathcal{D}_n^h\}$). We note that the strategy of directly modeling the statistical dependencies among all of the training data and the test case is a departure from most existing classification methods. Given a training corpus, most methods try to fit a model or learn a classifier that best explains the training data and use this learned model to classify test cases. This two-phase approach introduces a separation in information acquisition for training and testing; consequently, active information acquisition is limited either to real-time diagnosis or to training-time active learning and does not fully allow modeling of the joint statistics for the training and the test data. Directly modeling the dependency of the test label $t_*$ on the training and the test data as described in Equation 1 allows us to reason about next best information to observe by considering how posterior distributions changes with the acquisition of missing information. Assuming that we can compute predictive distributions as given in Equation 1, the next section describes how we can utilize such models to actively seek information.

## 3.1 Decision-Theoretic Selective Sampling

We are interested in selectively sampling unobserved information, either about the training set or the test case, in order to make a better prediction. If available budget allows for multiple observations, our the goal is to determine an optimal set of variables to observe. However, performing such nonmyopic analyses is prohibitively expensive for many active learning heuristics [7]. In practice, the selective sampling task is performed in a greedy manner. That is starting from an empty set, the algorithm selects one element at a time according to the active learning criterion. We note that Krause et al. [6] provides a detailed analysis of myopic and non-myopic strategies, and describes situations where losses in a greedy approach can be bounded. In this work, we adopt a greedy strategy.

The decision-theoretic selective sampling criterion we use estimates the values of acquiring information, which in turn can be used as a guiding principle in active learning. We can quantify such

value in terms of information gain. Intuitively, knowing one more bit of information may tighten a probability distribution over the class of the test case. On the other hand, observations are acquired at a price. By considering this reduction in uncertainty along with the cost of obtaining such information, we can formulate a selective sampling criterion.

Let us assume that we have a probabilistic model and appropriate inference procedures that would allow us to compute the conditional distribution of the test label $t_*$ given all the observed entities $\mathcal{D}^o$ (Equation 1). Then, such computations can be used determining the expected information gain. Expected information gain is formally defined as expected reduction in uncertainty over the $t_*$ as we observe more evidence. In order to balance the benefit of observing a feature/label with the cost of its observation, we use expected *return on information* (ROI) as a selection criteria that aims to maximize information gain per unit cost:

$$\mathcal{ROI} : \hat{d} = \arg \max_{d \in \mathcal{D}^h} \frac{H(t_*|\mathcal{D}^o) - \mathsf{E}_d[H(t_*|d \cup \mathcal{D}^o)]}{C(d)} \tag{2}$$

Here, $H(\cdot)$ denotes the entropy and $\mathsf{E}_d[\cdot]$ is the expectation with respect to the current model. Note, here $d$ can either be a feature value or a label and $C(\cdot)$ denotes the cost associated with observing information $d$. This strategy differs from the VOI criteria that aims to minimize total operational cost of the system. Unlike VOI, the proposed criteria does not require that the gain from selective sampling and the cost of observing observation have the same currency; consequently, ROI can be used more generally. Note, the proposed framework for active information acquisition easily extends to scenarios where the cost and the benefits of the system can be measure in a single currency and VOI can be applied. Also note that while the ROI formulation we introduces considers a single test, similar computations can be done for a larger set of test points by considering the joint entropy over the test labels. Without the introduction of assumptions of conditional independence that are not overly restrictive (described below) the joint formulation can be computed as the sum of the ROI evaluated for each of the test cases. We now describe how we can model the joint statistics among the training and the test cases simultaneously.

## 3.2 Modeling Joint Dependencies

Let us consider a probabilistic model to describe the joint dependencies among features and the label of an instance. If we denote the parameters of the model with $\boldsymbol{\lambda}$, then, given the training data, the classical approach in learning the model would attempt to find a *best* value $\hat{\boldsymbol{\lambda}}$ according to some optimization criterion. However, in our case we are interested in modeling joint dependencies among all of the data (both training or testing). Consequently, in our analysis, we consider the model parameters $\boldsymbol{\lambda}$ as a random variable over which we marginalize in order to generate a posterior predictive distribution. Formally, we rewrite Equation 1 as:

$$p(t_*|\mathbf{x}_*^o, \mathcal{D}^o) = \int_{\boldsymbol{\lambda}} p(t_*, \boldsymbol{\lambda}|\mathbf{x}_*^o, \mathcal{D}^o), \tag{3}$$

where the Bayesian treatment of $\boldsymbol{\lambda}$ allows us to marginalize over $\boldsymbol{\lambda}$ and model direct statistical dependencies between the different data points; consequently, we can determine how different features and labels directly affect the test prediction. Note that considering model parameters $\boldsymbol{\lambda}$ as random variables is consistent with principles of Bayesian modeling and is similar in spirit to prior research, such as [9] and [15].

In order to compute the integral in Equation (3), we need to characterize $p(t_*, \boldsymbol{\lambda}|\mathbf{x}_*^o, \mathcal{D}^o)$, which in turn defines a joint distribution over all of the data instances and the parameters $\boldsymbol{\lambda}$ of the model. First, we consider individual data instances and model the joint distribution of features and labels of the instance as a Markov Random Field (MRF)[2]. Then, assuming conditional independence between data points[3] given the model parameters, the joint distribution that includes all the instances and the parameters $\boldsymbol{\lambda}$ can be written as:

$$p(\mathcal{D}, \boldsymbol{\lambda}) \propto p(\boldsymbol{\lambda}) \prod_{i=1}^{n} \frac{1}{Z(\boldsymbol{\lambda})} \exp[\boldsymbol{\lambda}^T \boldsymbol{\phi}(\mathbf{x}_i, t_i)]$$

Here, $Z(\boldsymbol{\lambda})$ is the partition function that normalizes the distribution, $\boldsymbol{\lambda}$ are parameters of the model with a Gaussian prior $p(\boldsymbol{\lambda}) \sim \mathcal{N}(0, \boldsymbol{\Lambda})$. Also, $\boldsymbol{\phi}(\mathbf{x}, t) = [t, tx_1, .., tx_2, \boldsymbol{\phi}(\mathbf{x})]$ is the appended feature set and is in correspondence with the underlying undirected graphical model. In theory, the features can be functions of all the individual features of $\mathbf{x}$. However, we restrict ourselves to a Boltzmann machine that has individual and pairwise features only and corresponds to an undirected graphical model $\mathcal{G}_F = \{\mathcal{V}_F, \mathcal{E}_F\}$ where each node $\mathcal{V}_F$ corresponds to an individual feature and the edges in $\mathcal{E}_F$ between the nodes correspond to the pairwise features. A fully connected $\mathcal{G}_F$ graph can represent an arbitrary distribution. However, the computational complexity of situations involving large numbers of features may require pruning of the graph to achieve tractability.

Using Bayes rule and the conditional independence assumption, Equation 3 reduces to:

$$p(t_*|\mathbf{x}_*^o, \mathcal{D}^o) = \int_{\boldsymbol{\lambda}} p(t_*|\mathbf{x}_*^o, \boldsymbol{\lambda}) \cdot p(\boldsymbol{\lambda}|\mathcal{D}^o) \tag{4}$$

The first term $p(t_*|\mathbf{x}_*^o, \boldsymbol{\lambda})$ inside the integral can be interpreted as likelihood of $t_*$ given the observed components $\mathbf{x}_*$ of the test case and the parameter $\boldsymbol{\lambda}$. Similarly $p(\boldsymbol{\lambda}|\mathcal{D}^o)$ is the posterior distribution over parameter $\boldsymbol{\lambda}$ given all the observations in the training corpus. We review details of these computations below.

### 3.3 Computational Challenges

Given the set of all observations $\mathcal{D}^o$, we first seek to infer the posterior distribution $p(\boldsymbol{\lambda}|\mathcal{D}^o)$ which can be written as:

$$p(\boldsymbol{\lambda}|\mathcal{D}^o) \propto p(\boldsymbol{\lambda}) \prod_{i=1}^{n} \int_{\mathcal{D}_i^h} p(\mathcal{D}_i^o, \mathcal{D}_i^h|\boldsymbol{\lambda})$$

Computing the posterior is intractable as it is a product of the Gaussian prior with non-Gaussian data likelihood terms. In general, the problem of inferring model parameters in an undirected graphical model is a hard one. Welling and Parise [15] propose Bethe-Laplace approximation to infer model parameters for a Markov Random Field. In a similar spirit, we employ Laplace approximation that uses Bethe or a tree-structured approximation albeit with data that is partially observed. The idea behind Laplace approximation is to fit a Gaussian at the mode $\hat{\boldsymbol{\lambda}}$ of the exact posterior distribution $p(\boldsymbol{\lambda}|\mathcal{D}^o) \approx \mathcal{N}(\hat{\boldsymbol{\lambda}}, \Sigma)$, where:

$$\Sigma = \mathsf{E}_{\hat{\boldsymbol{\lambda}}}[\boldsymbol{\phi}(\mathbf{x}, t)\boldsymbol{\phi}(\mathbf{x}, t)^T] - \mathsf{E}_{\hat{\boldsymbol{\lambda}}}[\boldsymbol{\phi}(\mathbf{x}, t)]\mathsf{E}_{\hat{\boldsymbol{\lambda}}}[\boldsymbol{\phi}(\mathbf{x}, t)]^T$$

Here, $\mathsf{E}_{\hat{\boldsymbol{\lambda}}}[\cdot]$ denote expectation with respect to $p(\mathbf{x}, t|\hat{\boldsymbol{\lambda}})$. Note, that it is non-trivial to find the mode $\hat{\boldsymbol{\lambda}}$ as well as the covariance matrix $\Sigma$, as the underlying graphical structure is complex. While the covariance $\Sigma$ is approximated using the linear response algorithm [15], the mode $\hat{\boldsymbol{\lambda}}$ is usually found by running a gradient descent procedure that minimizes the negative log of the posterior ($\mathcal{L} = -\log(p(\boldsymbol{\lambda}|D))$). The gradients of this objective can be succinctly written as:

$$\nabla\mathcal{L} = \boldsymbol{\Lambda}^{-1}\boldsymbol{\lambda} - \sum_{i=1}^{n} [\mathsf{E}_{\boldsymbol{\lambda}, \mathcal{D}_i^o}[\boldsymbol{\phi}(\mathbf{x}, t)] - \mathsf{E}_{\boldsymbol{\lambda}}[\boldsymbol{\phi}(\mathbf{x}, t)]] \tag{5}$$

Here, $\mathsf{E}_{\boldsymbol{\lambda}, \mathcal{D}_i^o}[\cdot]$ is the expectation with respect to the distribution conditioned on the observed variables: $p(\mathbf{x}|\boldsymbol{\lambda}, \mathcal{D}_i^o)$. Note, that computing the first expectation term is trivial for the fully observed case. However, partially observed cases requires exact inference. Similarly, the computation of the second expectation term in the gradient requires exact inference. For the fully connected graphs, exact inference is hard and we must rely on approximations.

One approach is to approximate $\mathcal{G}_F$ by a tree, which we denote as $\mathcal{G}_{MI}$ that preserves an estimation of mutual information among variables. Specifically $\mathcal{G}_{MI}$ is the maximal spanning tree of an undirected graphical model, which has the same structure as the original graph and with edges weighted by empirical mutual information.

We have the choice of either running loopy belief propagation (BP) for approximate inference on the full graph $\mathcal{G}_F$ or doing an exact inference on the tree approximation $\mathcal{G}_{MI}$. As the features $\boldsymbol{\phi}(\mathbf{x}, t)$ only consist of single and pairwise variables, belief propagation directly provides the required expectations over the features of MRF. In our work, we observed better results when using loopy BP;

however, it was much faster to run inference on the tree structured graphs. Consequently, we used loopy BP to compute the posterior $p(\boldsymbol{\lambda}|D^o)$ given the training data. Also note that given the Gaussian approximation to $p(\boldsymbol{\lambda}|D^o)$, the required predictive distribution $p(t_*|\mathbf{x}_*, \mathcal{D}^o)$ can be computed using sampling [15]. Finally, ROI computations require that for each $d \in \mathcal{D}^h$, we infer $p(t_*|d \cup \mathcal{D}^o)$ for $d = 0$ and $d = 1$ and compute the expected conditional entropy. This repeated inference for all the missing bits in the data can be time consuming; thus, the tree-structured approximation was used to do all ROI computations and to determine the next bit of information to seek.

## 4   Experiments and Results

We shall compare proposed active information acquisition, which does not distinguish between induction-time and diagnosis-time analyses, against other alternatives on a synthetic dataset and two real-world applications. Previewing our results, we find that the proposed scheme outperforms its competitors in terms of accuracy over the test points and provides a significant boost for considerably less incurred cost. The significant gains we obtained over approaches that limit themselves to separately consider induction-time or diagnosis-time information acquisition suggests that the holistic perspective can provide broader and more efficient options to acquire information.

### 4.1   Experiments with Synthetic Data

We first sought to evaluate the basic operation of the proposed framework with a synthetic training set of Boolean data generated by randomly sampling labels with a fair coin toss. The features of the data are 14 dimensional and consist of partially informative and partially random features. Out of the 14 features, seven are randomly generated using a fair coin toss, while the rest of the features are generated by multiplying the label with all of the seven randomly generated features individually. We note that, even with full observations and a perfect data model for 0.78% of the cases, the prediction cannot be better than random. This arises whenever all of the randomly generated bits are 0 which in turn blocks any information about the label being observed. For the rest of the cases, perfect prediction is feasible with only seven features. We considered a dataset with 100 examples for experiments on this synthetic data. Further, we consider a 50-50 train and test split and assume that 25% of the total bits are unobserved and that the target of the selective sampling procedure is to determine the best next observations to make so as to best predict the labels for the test cases.

We assume that the cost of observing a label in the training data is directly proportional to the possible number of features that can be computed for every data point (that is, $c(d) = Dim$). The features, drawn from either the training or testing set, are much cheaper and have a unit cost of observation. We set the costs of observing labels of test cases to infinity; consequently the active learning methods never observe them.

We compared the joint selection (*Diagnosis+Induction*) advocated in this work with 1) diagnosis-time active information acquisition (*Diagnosis*), where information bits are sampled only from the test case at hand and 2) induction-time active acquisition (*Induction*). In addition, we considered two different flavors of induction-time active inquisition where either only features or only labels were allowed to be sampled. We refer to these two flavors as *Induction (features only)* and *Induction (labels only)* respectively. In all of the cases, we used ROI for active learning as described in section 3.1. Finally, we compare these methods with the baseline of random sampling strategy.

Figure 2 (left) shows the recognition results with increasing costs during active acquisition of information. We plot the overall classification accuracy over the test set on the $y$-axis and the cost incurred on the $x$-axis. Each point on the graph signifies an average recognition on the test set over 10 random training and test splits. From the figure, we see that all sampling strategies show increases in accuracy as the cost increases, but Diagnosis+Induction has advantages over other methods. First, Diagnosis+Induction obtains better recognition results for a fixed incurred cost, outperforming the diagnosis-time sampling strategy as well as all the flavors of induction-time information acquisition. Second, the Diagnosis+Induction sampling strategy levels off to the maximum performance fairly quickly when compared to other methods. Performance of Diagnosis only and Random sampling are noticeably worse than the other alternatives. Also, we note that the induction (features-only) stops abruptly for the synthetic case as most of the features in the learning problem are uninformative; after initial rounds the algorithm stops sampling. In summary, all of the active methods for active

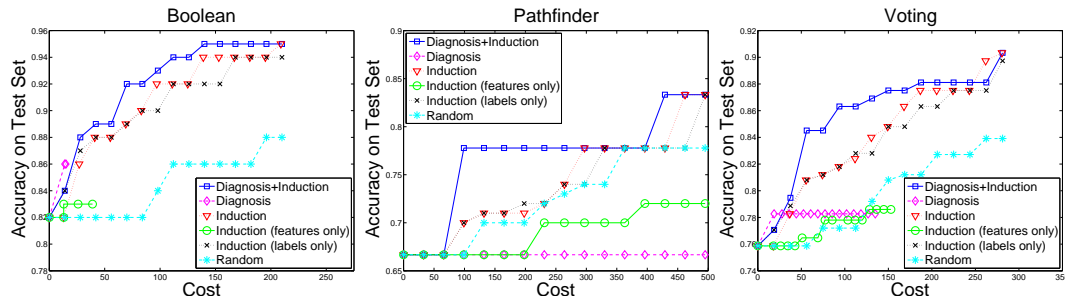

Figure 2: Comparison of various selective selection schemes (best viewed in color).

information acquisition do better than random; however, Induction+Diagnosis strategy achieves best combination of recognition performance and cost efficiency.

In order to analyze different sampling methods, we look at the sampling behavior of different active learning mechanisms. Figure 3 (left) illustrates the statistics of sampled information at the termination of the active learning procedure. The bars with different shades denote the sampling distribution amongst training labels, training features and the test features, which are generated by averaging over the 10 runs. While the Induction (features only), Induction (labels only) and diagnosis strategy just acquire labels, features for training data and features for the test cases respectively, the Diagnosis+Induction approaches show acquisition of information from different kinds of sources. We note that the random sampling strategy also samples from both labels and features; however, as indicated by Figure 2 (left) this strategy is not optimal as it does not take the cost structure into account. Diagnosis+Induction is the most flexible scheme and it aims to acquire information from all facets of the classification problem by properly considering gains in predictive power and balancing it with the cost of information acquisition.

## 4.2   Experiment on Pathfinder Data

Availability and access of large medical databases enables us to build better predictive models for various diagnostic purposes. While most efforts have focused on active data acquisition for diagnosis only [5], our framework promises a broader set of options to a diagnostician, where he can reason whether to perform additional tests on a patient or seek more information about the training set.

We analyze one such scenario where the goal is to build a predictive model that would guide surgical pathologists who study the lymphatic system with the diagnosis of lymph-node diseases. This dataset consists of labels of "benign" or "malignant" to lymph-node follicles from 48 subjects. The features signify sets of histological features viewed at low and high power under the microscope that an expert surgical pathologist believed could be informative to that label. The proposed holistic perspective on active learning supports the scenario where pathologists in pursuit of a diagnosis need to determine the next observations either from the test case at hand or consider querying for historical records in order to successfully label the lymph node (or, more generally, diagnose the disease). For this experiment, we consider random splits 30 training examples and 18 test cases and again assume that 25% of the total bits are unobserved. The experiment protocol is same as the one for synthetic data where we report results averaged over 10 runs and the test set is used to compare the recognition performance.

The results on the Pathfinder data are shown in Figure 2 (Middle). As before, $x$-axis and $y$-axis denote costs incurred and overall classification accuracy on the test data over 10 random training and test splits. Again we see that the Diagnosis+Induction performs better than the other methods and attains high accuracy at a fairly low cost. However, one difference in this experiment is the fact that Random sampling strategy outperforms active Diagnosis and active Induction (features only). This suggests that the labels in the training cases are highly informative when compared to the features. This in turn is reflected by the similar performance of Diagnosis+Induction with Induction and Induction (only) towards the end of active learning run. Upon further analysis, we found that Diagnosis+Induction, Induction and Induction (labels only) end up selecting similar training labels, consequently reaching similar performance towards the end. This further reinforces the validity of the hypothesis that the training labels are very informative. On analyzing the sampling behavior of different methods (Figure 3 (middle)) we again find that the Diagnosis+Induction approaches show acquisition of information from different kinds of sources. However, we also note that the proportion of sampled training labels is remarkably few and very similar for both Diagnosis+Induction

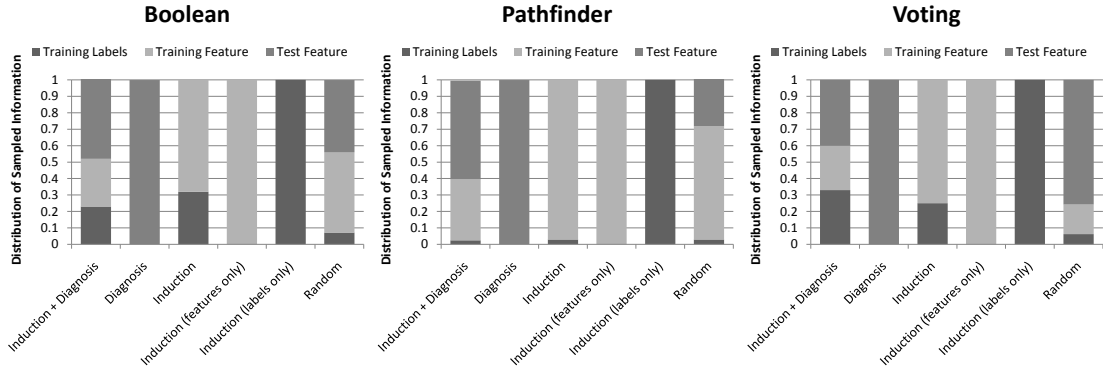

Figure 3: Statistics of different information selected in active learning.

and Induction, hinting that there might be particular cases that are highly informative about the prediction task. In summary, Diagnosis+Induction again provides best recognition rates at low costs, demonstrating the effectiveness of the unified perspective on active learning.

### 4.3 Experiments on Congressional Voting Records

Surveys have been popular information gathering tools, however, the cost of acquiring information by surveying can be costly and is often fraught with missing information. Intelligent information acquisition with active learning promises efficient use of limited resources. The holistic perspective on data acquisition can help avoid probing subjects for potentially risky or expensive questions by considering accessible information (for example, information such as demographics, age, etc.) or initially unavailable labels about the past survey takers.

We analyze a similar survey task of determining affiliation of subjects based on incomplete historical data. This data set includes votes for each of the U.S. House of Representatives Congressmen on the 16 key votes on United States policies. There are 435 data instances classified as Democrats versus Republican where 16 attributes for each of data represents a Yes or No on a vote. Further, out of $435 \times 16$ features, there are 392 instances missing. The presence of missing features makes this a challenging active-learning problem. We consider 10 random splits with 100 training instances and 335 test cases and report results averaged over these splits.

Experimental results on the voting data are shown in Figure 2 (right). Each point on the graph signifies an average recognition on the test set over 10 random training and test splits. Similar to the earlier experiments, we see improvement in recognition accuracy on the test set for different sampling schemes. Performance of Diagnosis only, Induction (features only), and Random sampling are noticeably worse than the other alternatives. Diagnosis+Induction again shows superior performance attaining a high accuracy at a relatively low cost. Upon analyzing the statistics of sampled information (Figure 3 (right)) at the termination of the active learning procedure, we see that while Diagnosis+Induction approaches show acquisition of information from different kinds of sources, it is significantly different from the Random strategy whose sampling distribution is close to the true distribution of the available information bits. By considering information gain and the cost structure through ROI, Diagnosis+Induction is able to achieve the best combination of recognition performance and cost efficiency.

## 5 Conclusion

We introduced a scheme for active data acquisition that removes the separation between diagnosis-time and induction-time active information acquisition. The task of diagnosis changes qualitatively with the use of methods that take a more holistic perspective on active learning, simultaneously considering information acquisition for extending a case library as well as for identifying the next best features to observe about the diagnostic challenge at hand. We ran several experiments that showed the effectiveness of combining diagnosis-time and induction-time active learning. We are pursuing several related challenges and opportunities, including analysis of approximate inference techniques and non-myopic extensions.

**References**

[1] N. Cesa-Bianchi, A. Conconi and C. Gentile (2003). Learning probabilistic linear-threshold classifiers via selective sampling. COLT.

[2] S. Dasgupta, A. T. Kalai and C. Monteleoni (2005). Analysis of perceptron-based active learning. COLT.

[3] Y. Freund, H. S. Seung, E. Shamir and N. Tishby (1997). Selective Sampling Using the Query by Committee Algorithm. Machine Learning Volume 28.

[4] R. Greiner, A. Grove and D. Roth (2002). Learning Cost-Sensitive Active Classifiers. Artificial Intelligence Volume 139(2).

[5] D. Heckerman, E. Horvitz, and B. N. Nathwani (1992). Toward Normative Expert Systems: Part I The Pathfinder Project. Methods of Information in Medicine 31:90-105.

[6] P. Kanani and P. Melville (2008). Prediction-time Active Feature-value Acquisition for Customer Targeting. NIPS Workshop on Cost Sensitive Learning.

[7] A. Krause, A. Singh and C. Guestrin (2008). Near-optimal Sensor Placements in Gaussian Processes: Theory, Efficient Algorithms and Empirical Studies. JMLR Volume 9(2).

[8] D. Lizotte, O. Madani and R. Greiner (2003). Budgeted Learning of Naive-Bayes Classifiers. UAI.

[9] D. MacKay (1992a). Information-Based Objective Functions for Active Data Selection. Neural Computation Volume 4(4).

[10] N. Roy and A. McCallum (2001). Toward Optimal Active Learning through Sampling Estimation of Error Reduction. ICML.

[11] M. Saar-Tschansky, P. Melville and F. Provost (2008). Active Feature-value Acquisition. Management Science.

[12] V. S. Sheng and C. X. Ling (2006). Feature value acquisition in testing: a sequential batch test algorithm. ICML.

[13] S. Tong and D. Koller (2001). Support Vector Machine Active Learning with Applications to Text Classification. JMLR Volume 2.

[14] S. Tong and D. Koller (2001). Active learning for parameter estimation in Bayesian networks. NIPS.

[15] M. Welling and S. Parise (2006) Bayesian Random Fields: The Bethe-Laplace Approximation. UAI.

## Footnotes

[1]For simplicity, we limit our discussion to a single test point; the analysis described generalizes directly to considering larger set of test points

[2]We limit ourselves to the case where both the labels and the features are binary (0 or 1).

[3]The conditional independence assumption also allows us to compute ROI for a set of test cases by summing individual ROI values.
